# Reinforcement Learning with Function Approximation Converges to a Region

**Geoffrey J. Gordon**
*ggordon@cs.cmu.edu*

## Abstract

Many algorithms for approximate reinforcement learning are not known to converge. In fact, there are counterexamples showing that the adjustable weights in some algorithms may oscillate within a region rather than converging to a point. This paper shows that, for two popular algorithms, such oscillation is the worst that can happen: the weights cannot diverge, but instead must converge to a bounded region. The algorithms are SARSA(0) and V(0); the latter algorithm was used in the well-known TD-Gammon program.

## 1  Introduction

Although there are convergent online algorithms (such as TD($\lambda$) [1]) for learning the parameters of a linear approximation to the value function of a Markov process, no way is known to extend these convergence proofs to the task of online approximation of either the state-value ($V^*$) or the action-value ($Q^*$) function of a general Markov decision process. In fact, there are known counterexamples to many proposed algorithms. For example, fitted value iteration can diverge even for Markov processes [2]; $Q$-learning with linear function approximators can diverge, even when the states are updated according to a fixed update policy [3]; and SARSA(0) can oscillate between multiple policies with different value functions [4].

Given the similarities between SARSA(0) and $Q$-learning, and between V(0) and value iteration, one might suppose that their convergence properties would be identical. That is not the case: while $Q$-learning can diverge for some exploration strategies, this paper proves that the iterates for trajectory-based SARSA(0) converge with probability 1 to a fixed region. Similarly, while value iteration can diverge for some exploration strategies, this paper proves that the iterates for trajectory-based V(0) converge with probability 1 to a fixed region.[1]

The question of the convergence behavior of SARSA($\lambda$) is one of the four open theoretical questions of reinforcement learning that Sutton [5] identifies as "particularly important, pressing, or opportune." This paper covers SARSA(0), and together

with an earlier paper [4] describes its convergence behavior: it is stable in the sense that there exist bounded regions which with probability 1 it eventually enters and never leaves, but for some Markov decision processes it may not converge to a single point. The proofs extend easily to SARSA($\lambda$) for $\lambda > 0$.

Unfortunately the bound given here is not of much use as a practical guarantee: it is loose enough that it provides little reason to believe that SARSA(0) and V(0) produce useful approximations to the state- and action-value functions. However, it is important for several reasons. First, it is the best result available for these two algorithms. Second, such a bound is often the first step towards proving stronger results. Finally, in practice it often happens that after some initial exploration period, only a few different policies are ever greedy; if this is the case, the strategy of this paper could be used to prove much tighter bounds.

Results similar to the ones presented here were developed independently in [6].

## 2 The algorithms

The SARSA(0) algorithm was first suggested in [7]. The V(0) algorithm was popularized by its use in the TD-Gammon backgammon playing program [8].[2]

Fix a Markov decision process $M$, with a finite set $S$ of states, a finite set $A$ of actions, a terminal state $T$, an initial distribution $S_0$ over $S$, a one-step reward function $r : S \times A \to R$, and a transition function $\delta : S \times A \to S \cup \{T\}$. ($M$ may also have a discount factor $\gamma$ specifying how to trade future rewards against present ones. Here we fix $\gamma = 1$, but our results carry through to $\gamma < 1$.) Both the transition and reward functions may be stochastic, so long as successive samples are independent (the Markov property) and the reward has bounded expectation and variance. We assume that all states in $S$ are reachable with positive probability.

We define a policy $\pi$ to be a function mapping states to probability distributions over actions. Given a policy we can sample a trajectory (a sequence of states, actions, and one-step rewards) by the following rule: begin by selecting a state $s_0$ according to $S_0$. Now choose an action $a_0$ according to $\pi(s_0)$. Now choose a one-step reward $r_0$ according to $r(s_0, a_0)$. Finally choose a new state $s_1$ according to $\delta(s_0, a_0)$. If $s_1 = T$, stop; otherwise repeat. We assume that all policies are proper, that is, that the agent reaches $T$ with probability 1 no matter what policy it follows. (This assumption is satisfied trivially if $\gamma < 1$.)

The reward for a trajectory is the sum of all of its one-step rewards. Our goal is to find an optimal policy, that is, a policy which on average generates trajectories with the highest possible reward. Define $Q^*(s, a)$ to be the best total expected reward that we can achieve by starting in state $s$, performing action $a$, and acting optimally afterwards. Define $V^*(s) = \max_a Q^*(s, a)$. Knowledge of either $Q^*$ or the combination of $V^*$, $\delta$, and $r$ is enough to determine an optimal policy.

The SARSA(0) algorithm maintains an approximation to $Q^*$. We will write $Q(s, a)$ for $s \in S$ and $a \in A$ to refer to this approximation. We will assume that $Q$ is a full-rank linear function of some parameters $w$. For convenience of notation, we will write $Q(T, a) = 0$ for all $a \in A$, and tack an arbitrary action onto the end of all trajectories (which would otherwise end with the terminal state). After seeing

a trajectory fragment $s, a, r, s', a'$, the SARSA(0) algorithm updates

$$Q(s, a) \leftarrow r + Q(s', a')$$

The notation $Q(s, a) \leftarrow V$ means that the parameters, $w$, which represent $Q(s, a)$ should be adjusted by gradient descent to reduce the error $(Q(s, a) - V)^2$; that is, for some preselected learning rate $\alpha \geq 0$,

$$w_{\text{new}} = w_{\text{old}} + \alpha(V - Q(s, a))\frac{\partial}{\partial w}Q(s, a)$$

For convenience, we assume that $\alpha$ remains constant within a single trajectory. We also make the standard assumption that the sequence of learning rates is fixed before the start of learning and satisfies $\sum_t \alpha_t = \infty$ and $\sum_t \alpha_t^2 < \infty$.

We will consider only the trajectory-based version of SARSA(0). This version changes policies only between trajectories. At the beginning of each trajectory, it selects the $\epsilon$-greedy policy for its current $Q$ function. From state $s$, the $\epsilon$-greedy policy chooses the action $\arg\max_a Q(s, a)$ with probability $1 - \epsilon$, and otherwise selects uniformly at random among all actions. This rule ensures that, no matter the sequence of learned $Q$ functions, each state-action pair will be visited infinitely often. (The use of $\epsilon$-greedy policies is not essential. We just need to be able to find a region that contains all of the approximate value functions for every policy considered, and a bound on the convergence rate of TD(0).)

We can compare the SARSA(0) update rule to the one for $Q$-learning:

$$Q(s, a) \leftarrow r + \max_b Q(s, b)$$

Often $a'$ in the SARSA(0) update rule will be the same as the maximizing $b$ in the $Q$-learning update rule; the difference only appears when the agent takes an exploring action, i.e., one which is not greedy for the current $Q$ function.

The V(0) algorithm maintains an approximation to $V^*$ which we will write $V(s)$ for all $s \in S$. Again, we will assume $V$ is a full-rank linear function of parameters $w$, and $V(T)$ is held fixed at 0. After seeing a trajectory fragment $s, a, r, s'$, V(0) sets

$$V(s) \leftarrow r + V(s')$$

This update ignores $a$. Often $a$ is chosen according to a greedy or $\epsilon$-greedy policy for a recent $V$. However, for our analysis we only need to assume that we consider finitely many policies and that the policy remains fixed during each trajectory.

We leave open the question of whether updates to $w$ happen immediately after each transition or only at the end of each trajectory. As pointed out in [9], this difference will not affect convergence: the updates within a single trajectory are $O(\alpha)$, so they cause a change in $Q(s, a)$ or $V(s)$ of $O(\alpha)$, which means subsequent updates are affected by at most $O(\alpha^2)$. Since $\alpha$ is decaying to zero, the $O(\alpha^2)$ terms can be neglected. (If we were to change policies during the trajectory, this argument would no longer hold, since small changes in $Q$ or $V$ can cause large changes in the policy.)

## 3    The result

Our result is that the weights $w$ in either SARSA(0) or V(0) converge with probability 1 to a fixed region. The proof of the result is based on the following intuition: while SARSA(0) and V(0) might consider many different policies over time, on any given trajectory they always follow the TD(0) update rule for some policy. The TD(0) update is, under general conditions, a 2-norm contraction, and so would

converge to its fixed point if it were applied repeatedly; what causes SARSA(0) and V(0) not to converge to a point is just that they consider different policies (and so take steps towards different fixed points) during different trajectories. Crucially, under general conditions, all of these fixed points are within some bounded region. So, we can view the SARSA(0) and V(0) update rules as contraction mappings plus a bounded amount of "slop." With this observation, standard convergence theorems show that the weight vectors generated by SARSA(0) and V(0) cannot diverge.

**Theorem 1** *For any Markov decision process $M$ satisfying our assumptions, there is a bounded region $R$ such that the SARSA(0) algorithm, when acting on $M$, produces a series of weight vectors which with probability 1 converges to $R$. Similarly, there is another bounded region $R'$ such that the V(0) algorithm acting on $M$ produces a series of weight vectors converging with probability 1 to $R'$.*

PROOF: Lemma 2, below, shows that both the SARSA(0) and V(0) updates can be written in the form

$$w_{t+1} = w_t - \alpha_t(A_t w_t - r_t + \epsilon_t)$$

where $A_t$ is positive definite, $\alpha_t$ is the current learning rate, $E(\epsilon_t) = 0$, $\text{Var}(\epsilon_t) \leq K(1 + \|w_t\|^2)$, and $A_t$ and $r_t$ depend only on the currently greedy policy. ($A_t$ and $r_t$ represent, in a manner described in the lemma, the transition probabilities and one-step costs which result from following the current policy. Of course, $w_t$, $A_t$, and $r_t$ will be different depending on whether we are following SARSA(0) or V(0).)

Since $A_t$ is positive definite, the SARSA(0) and V(0) updates are 2-norm contractions for small enough $\alpha_t$. So, if we kept the policy fixed rather than changing it at the beginning of each trajectory, standard results such as Lemma 1 below would guarantee convergence. The intuition is that we can define a nonnegative potential function $J(w)$ and show that, on average, the updates tend to decrease $J(w)$ as long as $\alpha_t$ is small enough and $J(w)$ starts out large enough compared to $\alpha_t$.

To apply Lemma 1 under the assumption that we keep the policy constant rather than changing it every trajectory, write $A_t = A$ and $r_t = r$ for all $t$, and write $w_\pi = A^{-1}r$. Let $\rho$ be the smallest eigenvalue of $A$ (which must be real and positive since $A$ is positive definite). Write $s_t = Aw_t - r + \epsilon_t$ for the update direction at step $t$. Then if we take $J(w) = \|w - w_\pi\|^2$,

$$
\begin{aligned}
E(\nabla J(w_t)^{\mathrm{T}} s_t | w_t) &= 2(w_t - w_\pi)^{\mathrm{T}}(Aw_t - r + E(\epsilon_t)) \\
&= 2(w_t - w_\pi)^{\mathrm{T}}(Aw_t - Aw_\pi) \\
&\geq 2\rho\|w_t - w_\pi\|^2 \\
&= 2\rho J(w_t)
\end{aligned}
$$

so that $-s_t$ is a descent direction in the sense required by the lemma. It is easy to check the lemma's variance condition. So, Lemma 1 shows that $J(w_t)$ converges with probability 1 to 0, which means $w_t$ must converge with probability 1 to $w_\pi$.

If we pick an arbitrary vector $u$ and define $H(w) = \max(0, \|w - u\| - C)^2$ for a sufficiently large constant $C$, then the same argument reaches the weaker conclusion that $w_t$ must converge with probability 1 to a sphere of radius $C$ centered at $u$. To see why, note that $-s_t$ is also a descent direction for $H(w)$: inside the sphere, $H = 0$ and $\nabla H = 0$, so the descent condition is satisfied trivially. Outside the sphere,

$$
\begin{aligned}
\nabla H(w) &= 2(w - u)\frac{\|w - u\| - C}{\|w - u\|} \equiv d(w)(w - u) \\
\nabla H(w_t)^{\mathrm{T}} E(s_t | w_t) &= d(w_t)(w_t - u)^{\mathrm{T}} E(s_t | w_t)
\end{aligned}
$$

$$
\begin{aligned}
&= d(w_t)(w_t - w_\pi + w_\pi - u)^{\mathrm{T}} A(w_t - w_\pi) \\
&\geq d(w_t)(\rho \|w_t - w_\pi\|^2 - \|w_\pi - u\|\,\|A\|\,\|w_t - w_\pi\|)
\end{aligned}
$$

The positive term will be larger than the negative one if $\|w_t - w_\pi\|$ is large enough. So, if we choose $C$ large enough, the descent condition will be satisfied. The variance condition is again easy to check. Lemma 3 shows that $\nabla H$ is Lipschitz. So, Lemma 1 shows that $H(w_t)$ converges with probability 1 to 0, which means that $w_t$ must converge with probability 1 to the sphere of radius $C$ centered at $u$.

But now we are done: since there are finitely many policies that SARSA(0) or V(0) can consider, we can pick any $u$ and then choose a $C$ large enough that the above argument holds for all policies simultaneously. With this choice of $C$ the update for any policy decreases $H(w_t)$ on average as long as $\alpha_t$ is small enough, so the update for SARSA(0) or V(0) does too, and Lemma 1 applies. $\qquad\square$

The following lemma is Corollary 1 of [10]. In the statement of the lemma, a Lipschitz continuous function $F$ is one for which there exists a constant $L$ so that $\|F(u) - F(w)\| \leq L\|u - w\|$ for all $u$ and $w$. The Lipschitz condition is essentially a uniform bound on the derivative of $F$.

**Lemma 1** *Let $J$ be a differentiable function, bounded below by $J^*$, and let $\nabla J$ be Lipschitz continuous. Suppose the sequence $w_t$ satisfies*

$$
w_{t+1} = w_t - \alpha_t s_t
$$

*for random vectors $s_t$ independent of $w_{t+1}, w_{t+2}, \ldots$. Suppose $-s_t$ is a descent direction for $J$ in the sense that $E(s_t | w_t)^{\mathrm{T}} \nabla J(w_t) > \delta(\epsilon) > 0$ whenever $J(w_t) > J^* + \epsilon$. Suppose also that*

$$
E(\|s_t\|^2 | w_t) \leq K_1 J(w_t) + K_2 E(s_t | w_t)^{\mathrm{T}} \nabla J(w_t) + K_3
$$

*and finally that the constants $\alpha_t$ satisfy*

$$
\alpha_t > 0 \qquad \sum_t \alpha_t = \infty \qquad \sum_t \alpha_t^2 < \infty
$$

*Then $J(w_t) \to J^*$ with probability 1.*

Most of the work in proving the next lemma is already present in [1]. The transformation from an MDP under a fixed policy to a Markov chain is standard.

**Lemma 2** *The update made by SARSA(0) or V(0) during a single trajectory can be written in the form*

$$
w_{\mathrm{new}} = w_{\mathrm{old}} - \alpha(A_\pi w_{\mathrm{old}} - r_\pi + \epsilon)
$$

*where the constant matrix $A_\pi$ and constant vector $r_\pi$ depend on the currently greedy policy $\pi$, $\alpha$ is the current learning rate, and $E(\epsilon) = 0$. Furthermore, $A_\pi$ is positive definite, and there is a constant $K$ such that $\mathrm{Var}(\epsilon) \leq K(1 + \|w\|^2)$.*

PROOF: Consider the following Markov process $M_\pi$: $M_\pi$ has one state for each state-action pair in $M$. If $M$ has a transition which goes from state $s$ under action $a$ with reward $r$ to state $s'$ with probability $p$, then $M_\pi$ has a transition from state $\langle s, a \rangle$ with reward $r$ to state $\langle s', a' \rangle$ for every $a'$; the probability of this transition is $p\pi(a'|s')$. We will represent the value function for $M_\pi$ in the same way that we represented the $Q$ function for $M$; in other words, the representation for $V(\langle s, a \rangle)$ is the same as the representation for $Q(s, a)$. With these definitions, it is easy to see that TD(0) acting on $M_\pi$ produces exactly the same sequence of parameter changes

as SARSA(0) acting on $M$ under the fixed policy $\pi$. (And since $\pi(a|s) > 0$, every state of $M_\pi$ will be visited infinitely often.)

Write $T_\pi$ for the transition probability matrix of the above Markov process. That is, the entry of $T_\pi$ in row $\langle s, a \rangle$ and column $\langle s', a' \rangle$ will be equal to the probability of taking a step to $\langle s', a' \rangle$ given that we start in $\langle s, a \rangle$. By definition, $T_\pi$ is substochastic. That is, it has nonnegative entries, and its row sums are less than or equal to 1. Write $s$ for the vector whose $\langle s, a \rangle$th element is $S_0(s)\pi(a|s)$, that is, the probability that we start in state $s$ and take action $a$. Write $d_\pi = (I - T_\pi^{\mathrm{T}})^{-1}s$, where $I$ is the identity matrix. As demonstrated in, e.g., [11], $d_\pi$ is the vector of expected visitation frequencies under $\pi$; that is, the element of $d_\pi$ corresponding to state $s$ and action $a$ is the expected number of times that the agent will visit state $s$ and select action $a$ during a single trajectory following policy $\pi$. Write $D_\pi$ for the diagonal matrix with $d_\pi$ on its diagonal. Write $r$ for the vector of expected rewards; that is, the component of $r$ corresponding to state $s$ and action $a$ is $E(r(s, a))$. Finally write $X$ for the Jacobian matrix $\frac{\partial Q}{\partial w}$.

With this notation, Sutton [1] showed that the expected TD(0) update is

$$E(w_{\text{new}}|w_{\text{old}}) = w_{\text{old}} - \alpha X^{\mathrm{T}} D_\pi (I - T_\pi) X w_{\text{old}} + \alpha X^{\mathrm{T}} D_\pi r$$

(Actually, he only considered the case where all rewards are zero except on transitions from nonterminal to terminal states, but his argument works equally well for the more general case where nonzero rewards are allowed everywhere.) So, we can take $A_\pi = X^{\mathrm{T}} D_\pi (I - T_\pi) X$ and $r_\pi = X^{\mathrm{T}} D_\pi r$ to make $E(\epsilon) = 0$.

Furthermore, Sutton showed that, as long as the agent reaches the terminal state with probability 1 (in other words, as long as $\pi$ is proper) and as long as every state is visited with positive probability (which is true since all states are reachable and $\pi$ has a nonzero probability of choosing every action), the matrix $D_\pi(I - T_\pi)$ is strictly positive definite. Therefore, so is $A_\pi$.

Finally, as can be seen from Sutton's equations on p. 25, there are two sources of variance in the update direction: variation in the number of times each transition is visited, and variation in the one-step rewards. The visitation frequencies and the one-step rewards both have bounded variance, and are independent of one another. They enter into the overall update in two ways: there is one set of terms which is bilinear in the one-step rewards and the visitation frequencies, and there is another set of terms which is bilinear in the visitation frequencies and the weights $w$. The former set of terms has constant variance. Because the policy is fixed, $w$ is independent of the visitation frequencies, and so the latter set of terms has variance proportional to $\|w\|^2$. So, there is a constant $K$ such that the total variance in $\epsilon$ can be bounded by $K(1 + \|w\|^2)$.

A similar but simpler argument applies to V(0). In this case we define $M_\pi$ to have the same states as $M$, and to have the transition matrix $T_\pi$ whose element $s, s'$ is the probability of landing in $s'$ in $M$ on step $t + 1$, given that we start in $s$ at step $t$ and follow $\pi$. Write $s$ for the vector of starting probabilities, that is, $s_x = S_0(x)$. Now define $X = \frac{\partial V}{\partial w}$ and $d_\pi = (I - T_\pi^{\mathrm{T}})^{-1}s$. Since we have assumed that all policies are proper and that every policy considered has a positive probability of reaching any state, the update matrix $A_\pi = X^{\mathrm{T}} D_\pi (I - T_\pi) X$ is strictly positive definite. $\square$

**Lemma 3** *The gradient of the function* $H(w) = \max(0, \|w\| - 1)^2$ *is Lipschitz continuous.*

PROOF: Inside the unit sphere, $H$ and all of its derivatives are uniformly zero. Outside, we have

$$\nabla H = wd(w)$$

where $d(w) = \frac{\|w\|-1}{\|w\|}$, and

$$
\begin{aligned}
\nabla^2 H &= d(w)I + \nabla d(w)w^{\mathrm{T}} \\
&= d(w)I + \frac{w}{\|w\|^2}\frac{1}{\|w\|}w^{\mathrm{T}} \\
&= d(w)I + \frac{ww^{\mathrm{T}}}{\|w\|^2}(1 - d(w))
\end{aligned}
$$

The norm of the first term is $d(w)$, the norm of the second is $1 - d(w)$, and since one of the terms is a multiple of $I$ the norms add. So, the norm of $\nabla^2 H$ is 0 inside the unit sphere and 1 outside. At the boundary of the unit sphere, $\nabla H$ is continuous, and its directional derivatives from every direction are bounded by the argument above. So, $\nabla H$ is Lipschitz continuous. □

### Acknowledgements

Thanks to Andrew Moore and to the anonymous reviewers for helpful comments. This work was supported in part by DARPA contract number F30602–97–1–0215, and in part by NSF KDI award number DMS–9873442. The opinions and conclusions are the author's and do not reflect those of the US government or its agencies.

## Footnotes

[1]In a "trajectory-based" algorithm, the exploration policy may not change within a single episode of learning. The policy may change between episodes, and the value function may change within a single episode. (Episodes end when the agent enters a terminal state. This paper considers only episodic tasks, but since any discounted task can be transformed into an equivalent episodic task, the algorithms apply to non-episodic tasks as well.)

[2] The proof given here does not cover the TD-Gammon program, since TD-Gammon uses a nonlinear function approximator to represent its value function. Interestingly, though, the proof extends easily to cover games such as backgammon in addition to MDPs. It also extends to cover SARSA($\lambda$) and V($\lambda$) for $\lambda > 0$.

### References

[1] R. S. Sutton. Learning to predict by the methods of temporal differences. *Machine Learning*, 3(1):9–44, 1988.

[2] Geoffrey J. Gordon. Stable function approximation in dynamic programming. Technical Report CMU-CS-95-103, Carnegie Mellon University, 1995.

[3] L. C. Baird. Residual algorithms: Reinforcement learning with function approximation. In *Machine Learning: proceedings of the twelfth international conference*, San Francisco, CA, 1995. Morgan Kaufmann.

[4] Geoffrey J. Gordon. Chattering in SARSA($\lambda$). Internal report, 1996. CMU Learning Lab. Available from www.cs.cmu.edu/~ggordon.

[5] R. S. Sutton. Open theoretical questions in reinforcement learning. In P. Fischer and H. U. Simon, editors, *Computational Learning Theory (Proceedings of EuroCOLT'99)*, pages 11–17, 1999.

[6] D. P. de Farias and B. Van Roy. On the existence of fixed points for approximate value iteration and temporal-difference learning. *Journal of Optimization Theory and Applications*, 105(3), 2000.

[7] Gavin A. Rummery and Mahesan Niranjan. On-line Q-learning using connectionist systems. Technical Report 166, Cambridge University Engineering Department, 1994.

[8] G. Tesauro. TD-Gammon, a self-teaching backgammon program, achieves master-level play. *Neural Computation*, 6:215–219, 1994.

[9] T. Jaakkola, M. I. Jordan, and S. P. Singh. On the convergence of stochastic iterative dynamic programming algorithms. *Neural Computation*, 6:1185–1201, 1994.

[10] B. T. Polyak and Ya. Z. Tsypkin. Pseudogradient adaptation and training algorithms. *Automation and Remote Control*, 34(3):377–397, 1973. Translated from *Avtomatika i Telemekhanika*.

[11] J. G. Kemeny and J. L. Snell. *Finite Markov Chains*. Van Nostrand—Reinhold, New York, 1960.
